# A Large-Scale Human-Centric Benchmark for Referring Expression Comprehension in the LMM Era

**Fangyun Wei**\*
The University of Sydney
fwei8714@uni.sydney.edu.au

**Jinjing Zhao**\*
The University of Sydney
jzha0100@uni.sydney.edu.au

**Kun Yan**
Peking University
kyan2018@pku.edu.cn

**Hongyang Zhang**
University of Waterloo
hongyang.zhang@uwaterloo.ca

**Chang Xu**
The University of Sydney
c.xu@sydney.edu.au

## Abstract

Prior research in human-centric AI has primarily addressed single-modality tasks like pedestrian detection, action recognition, and pose estimation. However, the emergence of large multimodal models (LMMs) such as GPT-4V has redirected attention towards integrating language with visual content. Referring expression comprehension (REC) represents a prime example of this multimodal approach. Current human-centric REC benchmarks, typically sourced from general datasets, fall short in the LMM era due to their limitations, such as insufficient testing samples, overly concise referring expressions, and limited vocabulary, making them inadequate for evaluating the full capabilities of modern REC models. In response, we present HC-RefLoCo (Human-Centric Referring Expression Comprehension with Long Context), a benchmark that includes 13,452 images, 24,129 instances, and 44,738 detailed annotations, encompassing a vocabulary of 18,681 words. Each annotation, meticulously reviewed for accuracy, averages 93.2 words and includes topics such as appearance, human-object interaction, location, action, celebrity, and OCR. HC-RefLoCo provides a wider range of instance scales and diverse evaluation protocols, encompassing accuracy with various IoU criteria, scale-aware evaluation, and subject-specific assessments. Our experiments, which assess 24 models, highlight HC-RefLoCo's potential to advance human-centric AI by challenging contemporary REC models with comprehensive and varied data. Our benchmark, along with the evaluation code, are available at https://github.com/ZhaoJingjing713/HC-RefLoCo.

## 1 Introduction

Prior research in human-centric AI has predominantly concentrated on single-modality algorithms tasked with understanding, interacting with, or analyzing human behaviors and features. These tasks include face detection [93, 10, 32, 69, 81, 94, 53] and recognition [63, 68, 52, 25, 24], pedestrian detection [76, 95, 97, 9] and re-identification [19, 49, 12, 42, 86], action recognition [34, 14, 45, 51, 82], pose estimation [66, 58, 91, 77], among others. However, the recent emergence of large multimodal models (LMMs) such as GPT-4V [54–56] and Google Gemini [70] has shifted the research landscape towards integrating language semantics with visual content. This paradigm shift heralds a new era in human-centric AI, emphasizing multimodality. Referring expression comprehension (REC) [83, 67, 22, 40, 44, 90, 37, 47, 48, 99, 100, 43] is a prime example of such a

---

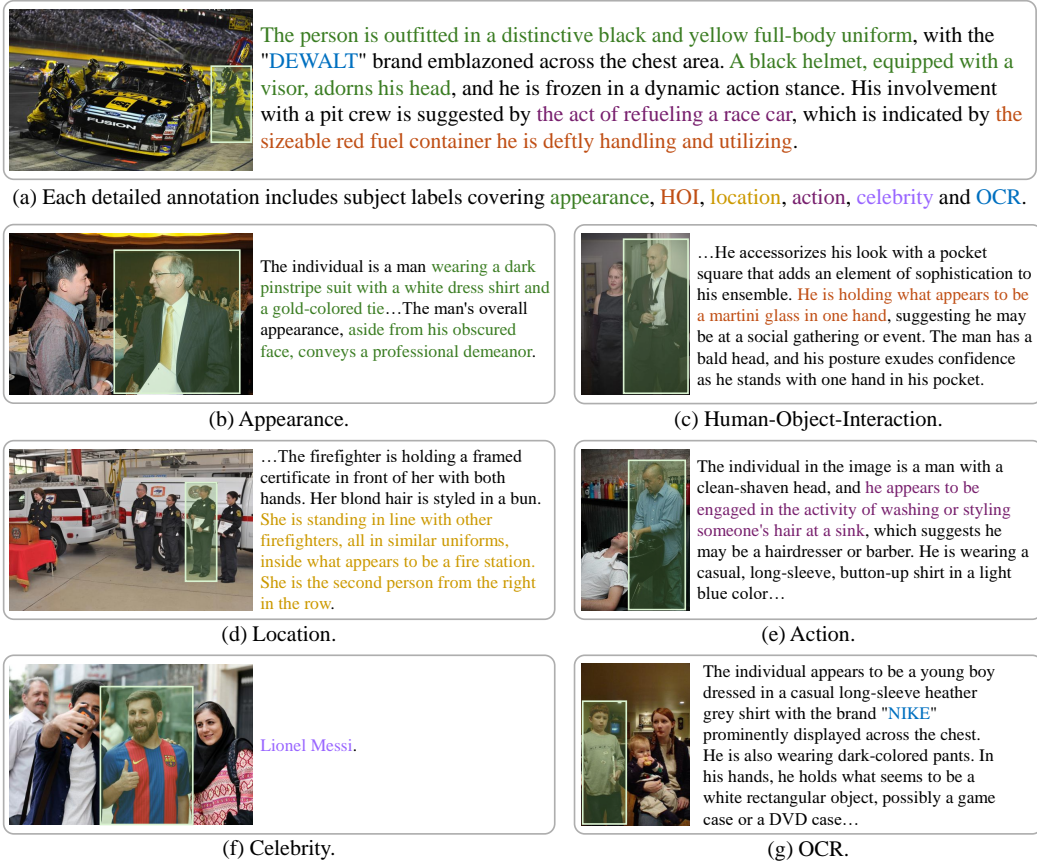

(a) Each detailed annotation includes subject labels covering appearance, HOI, location, action, celebrity and OCR.

The person is outfitted in a distinctive black and yellow full-body uniform, with the "DEWALT" brand emblazoned across the chest area. A black helmet, equipped with a visor, adorns his head, and he is frozen in a dynamic action stance. His involvement with a pit crew is suggested by the act of refueling a race car, which is indicated by the sizeable red fuel container he is deftly handling and utilizing.

The individual is a man wearing a dark pinstripe suit with a white dress shirt and a gold-colored tie…The man's overall appearance, aside from his obscured face, conveys a professional demeanor.

(b) Appearance.

…He accessorizes his look with a pocket square that adds an element of sophistication to his ensemble. He is holding what appears to be a martini glass in one hand, suggesting he may be at a social gathering or event. The man has a bald head, and his posture exudes confidence as he stands with one hand in his pocket.

(c) Human-Object-Interaction.

…The firefighter is holding a framed certificate in front of her with both hands. Her blond hair is styled in a bun. She is standing in line with other firefighters, all in similar uniforms, inside what appears to be a fire station. She is the second person from the right in the row.

(d) Location.

The individual in the image is a man with a clean-shaven head, and he appears to be engaged in the activity of washing or styling someone's hair at a sink, which suggests he may be a hairdresser or barber. He is wearing a casual, long-sleeve, button-up shirt in a light blue color…

(e) Action.

Lionel Messi.

(f) Celebrity.

The individual appears to be a young boy dressed in a casual long-sleeve heather grey shirt with the brand "NIKE" prominently displayed across the chest. He is also wearing dark-colored pants. In his hands, he holds what seems to be a white rectangular object, possibly a game case or a DVD case…

(g) OCR.

Figure 1: (a) An Example from our HC-RefLoCo benchmark. For each target object, we provide a comprehensive and detailed text description, with an average length of 93.2 words. Each sentence within this description is classified into one of the following categories: (b) appearance, (c) human-object interaction, (d) location, (e) action, (f) celebrity, (g) optical character recognition, or None.

multimodal task. REC involves the localization of specific instances described by natural language inputs. Despite its relevance, there is a notable lack of benchmarks specifically designed to evaluate REC in human-centered contexts. This paper aims to address this gap by developing benchmarks for human-centric REC in the era of large multimodal models [33, 20, 36, 41, 101, 6, 2, 1, 39, 88, 62, 59].

Existing human-centric REC benchmarks primarily derive from general REC datasets, such as RefCOCO [23], RefCOCO+ [23], and RefCOCOg [50], by filtering images and text annotations related to human categories. These resulting benchmarks, termed HC-RefCOCO, HC-RefCOCO+, and HC-RefCOCOg, where "HC" stands for human-centric, typically include a limited number of test samples, as illustrated in Table 1. For instance, HC-RefCOCO comprises only 1,519 images with 10,771 text annotations. Moreover, these benchmarks utilize brief text descriptions of the target instances, with the average word count of annotations being 3.4, 3.3, and 8.9 for HC-RefCOCO, HC-RefCOCO+, and HC-RefCOCOg, respectively. The advent of large language models, such as GPT-4 [54] and LLAMA [71], has significantly enhanced the language understanding capabilities of AI models, making the processing of short texts less challenging in the REC task. Consequently, there is a pressing need for more comprehensive and challenging benchmarks that reflect the advanced capabilities of contemporary AI models in human-centric REC.

In this work, we introduce a new benchmark called HC-RefLoCo, which stands for Human-Centric Referring Expression Comprehension with Long Context. Comprehensive statistics can be found in Table 1. Our benchmark is characterized by the following five features:

**Large Scale.** Our benchmark includes 13,452 images accompanied by 24,129 instances with 44,738 annotations (referring expressions), providing a substantial dataset for HC-REC testing.

Table 1: Comparison between human-centric (HC) referring expression comprehension benchmarks and the proposed HC-RefLoCo benchmark. Statistics for HC-RefCOCO, HC-RefCOCO+, and HC-RefCOCOg are derived from the combination of their respective validation and test sets. Vocab.: vocabulary. Avg.: average.

| Dataset | Images | Instances | Annotations | Avg. Words | Vocab. | Instance Size | Subjects |
|---|---|---|---|---|---|---|---|
| HC-RefCOCO [23] | 1,519 | 3,754 | 10,771 | 3.4 | 2,251 | 114.0 - 603.2 | - |
| HC-RefCOCO+ [23] | 1,519 | 3,754 | 10,908 | 3.3 | 2,702 | 114.0 - 603.2 | - |
| HC-RefCOCOg [50] | 1,521 | 2,669 | 5,253 | 8.9 | 2,891 | 89.7 - 610.5 | - |
| HC-RefLoCo (Ours) | 13,452 | 24,129 | 44,738 | 93.2 | 18,681 | 62.5 - 3720.7 | 6 |

**Long and Detailed Descriptions.** Utilizing the cutting-edge language model, GPT-4, we generate extensive descriptions (annotations) for each target instance. Each annotation is meticulously *reviewed* and *manually* revised to correct any hallucination issues. This rigorous process ensures the accuracy of the benchmark. The descriptions vary in length from 15 to 241 words, averaging 93.2 words. They encompass an extensive vocabulary of 18,681 words. An example can be found in Figure 1(a).

**Subject Labels.** Each text annotation is an extensive paragraph comprising multiple sentences. We manually categorize each sentence into one of the following subjects: appearance, human-object interaction (HOI), location, action, celebrity, optical character recognition (OCR), or None, as illustrated in Figure 1. This detailed labeling process enables a focused evaluation of modern REC models, assessing their proficiency in interpreting and processing varied linguistic inputs associated with specific subjects. By incorporating a wide range of subjects, we aim to provide a robust benchmark that challenges and advances the current capabilities of REC models, ensuring they can adeptly handle the complexity and diversity present in real-world applications.

**Broader Coverage of Instance Scales.** Compared to existing HC-REC benchmarks like HC-RefCOCO, HC-RefCOCO+, and HC-RefCOCOg, our dataset spans a wider range of instance scales. The square root of the instance size varies from 62.5 to 3720.7, with an average of 313.8, providing a more comprehensive representation of different object sizes.

**Various Evaluation Protocols.** Accuracy is a commonly used metric in evaluating existing REC models. An instance is considered successfully located if the Intersection over Union (IoU) between the predicted bounding box and the ground truth exceeds 0.5. This standard evaluation metric is referred to as $Acc_{0.5}$. To provide a comprehensive understanding of the models' strengths and weaknesses, we introduce various evaluation protocols, including:

- Accuracy using different IoU criteria such as $Acc_{0.5}$, $Acc_{0.75}$, $Acc_{0.9}$, and mean Accuracy (mAcc) across all IoU criteria, to thoroughly assess the models' localization capabilities.
- Accuracy on small, medium, and large instances to evaluate the models' efficiency across varying instance sizes.
- Subject-specific evaluation, which involves assessing models based on distinct subjects to validate the performance of REC models in managing diverse linguistic inputs and correlating detailed descriptions with precise visual elements.

In our experiments, we evaluate a total of 24 training-unconstrained models, which utilize any available data for training, including GPT-4V, bounding box-output models, and mask-output models, employing a range of evaluation protocols. With its extensive test samples, detailed annotations, the incorporation of subject labels, the broad coverage of instance scales, and the introduction of diverse evaluation protocols, we hope our benchmark will advance research in human-centric AI.

## 2 Related Work

**REC Benchmarks.** Referring expression comprehension (REC) refers to the process of localizing the specific instances described by natural language inputs. Current human-centric REC benchmarks primarily originate from general REC datasets like RefCOCO [23], RefCOCO+ [23], and RefCOCOg [50]. RefCOCO, which stems from the COCO2014 [38] dataset, contains 50,000 annotations across 19,994 images. The expressions in this benchmark are typically short and concise,

including many locational descriptions such as "Right guy", "Far left man", and "Guy on left". In contrast, RefCOCO+ is of a similar scale, with 49,856 annotations across 19,992 images, but intentionally omits locational prepositions like "left" and "right", thereby increasing semantic complexity. Examples include "Man with light hat" and "Guy in white". RefCOCOg, on the other hand, offers more extensive annotations compared to its predecessors, with examples like "A person in a hat on a wooden bench" and "A man in white playing Frisbee". To improve performance on these benchmarks, datasets like GRIT [59], GranD [62] and RecapD [17] are commonly utilized as training sources. Although not exclusively designed for REC, datasets like Flickr30k Entities [60, 89] and Visual Genome [29] are also frequently used for training. Compared to previous testing benchmarks, our HC-RefLoCo provides longer expressions, with an average length of 93.2 words, covering a vast vocabulary of 18,681 words.

**LMMs for Visual Grounding.** Recent advancements in large multimodal models (LMMs) such as Flamingo [1], BLIP-2 [33], MiniGPT-4 [101, 4], InstructBLIP [4], mPLUG-Owl [87], and LLaVA [41], have significantly enhanced the integration of vision and language modalities by leveraging the progress in large language models (LLMs) [54, 55, 70–72, 8]. These models have shown remarkable improvements in tasks related to image understanding and visual question answering. However, instance localization remains a challenging aspect that requires LMMs to not only comprehend the relationship between visual elements and language but also to accurately generate bounding boxes for target instances. The REC task serves as a critical benchmark to evaluate the localization capabilities of these models. Pioneering models like KOSMOS-2 [59], Shikra [5], Grounding-GPT [36], Qwen-VL [3], and the SPHINX series [39, 15] typically employ an auto-regressive causal Transformer with tokenized bounding box representations to tackle the REC task. To achieve more precise representations of target instances, recent approaches have suggested the use of masks instead of bounding boxes as outputs. Models such as LISA [30, 84], PixelLLM [98], PSALM [96], and GlaMM [62] extend the segmentation paradigm initially developed by SAM [26]. Another closely related area is open-vocabulary object detection and segmentation [16, 11, 79, 80], which aims to locate any objects and identify their class labels using a word or phrase. However, this task still differs from the REC task, where the models have to identify the target based on an extended text description rather than a single word or phrase.

# 3    Benchmark Construction and Analysis

## 3.1    Benchmark Construction

**Data Sources and Pre-Processing.** Our HC-RefLoCo benchmark is derived from several public object detection datasets, including the validation sets of COCO 2017 [38] and Objects365 [65], as well as the validation and testing sets of OpenImage v7 [28]. For COCO 2017 and Objects365, we retain all instances labeled as "person", whereas for OpenImage v7, we keep instances labeled as "human". We also exclude extremely small instances, specifically those occupying less than 1% of the total image area. We adopt the original bounding box annotations in these datasets. We also collect images of 367 celebrities from the LAION-5B [64] dataset. Each image in the dataset contains at least one of these celebrities and includes at least two people. The bounding boxes for the celebrities are manually annotated. Consequently, we compile a total of 3,520 images, each containing a single annotated instance.

In conclusion, our HC-RefLoCo benchmark comprises 200 images with 419 instances from COCO, 4,772 images with 10,070 instances from Objects365, 4,960 images with 10,120 instances from OpenImage v7, and 3,520 images with the same number of instances from LAION-5B.

**Referring Expression Generation.** Figure 2 illustrates the procedure for generating a referring expression (a.k.a. description) for each target instance. Given a target instance and its corresponding image, this involves a three-step process:

1. Employing GPT-4V to generate an instance-level description by inputting the cropped instance, following the prompt outlined in Section A.1.

2. Feeding the raw image into GPT-4V to expand the initial description generated in Step.1 by incorporating the context around the target instance, using the prompt described in Section A.2.

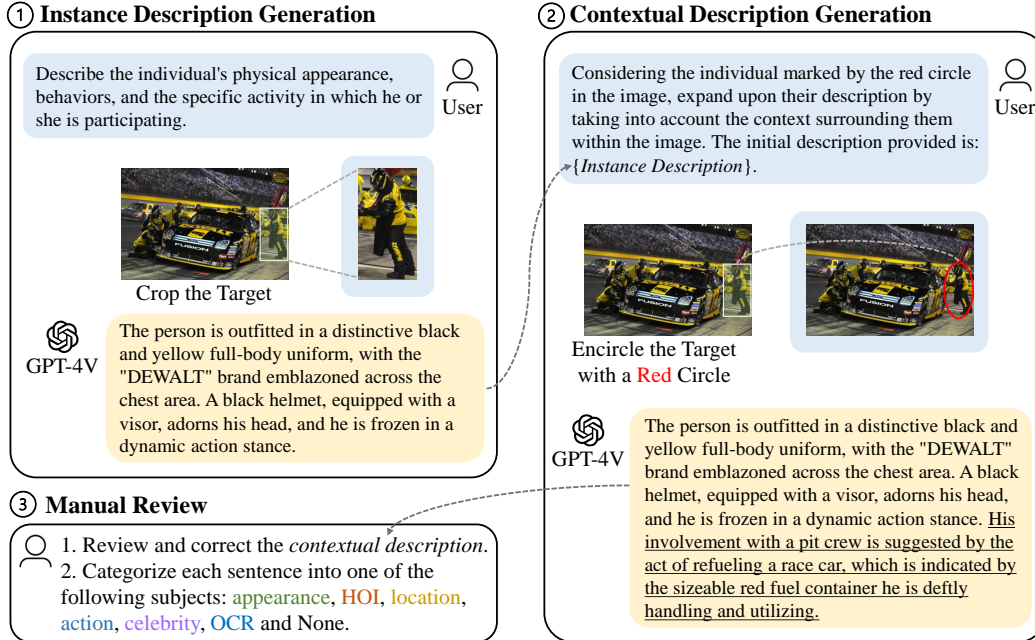

Figure 2: The process of generating a referring expression for each target instance. Inspired by recent studies on GPT-4V [85], which demonstrate that GPT-4V can pay more attention to instances highlighted by a red circle within an image, we similarly encircle the target instance in red in Step-2.

3. Subsequently, we manually review each referring expression to correct any errors, particularly those arising from hallucination issues in GPT-4V, ensuring that the descriptions accurately and uniquely identify the target instances.

**Annotation Expansion.** Up to now, our benchmark includes 13,452 images with 24,129 instances, each accompanied by a referring expression (annotation). Leveraging GPT-4's exceptional language capabilities, we prompt it to rewrite each referring expression, using the prompt detailed in Section A.3. This process effectively doubles the annotations. We then conduct a manual review of each rewritten referring expression, eliminating those that are improper or ambiguous to ensure that the revised annotations uniquely describe their respective target instances. Consequently, our final benchmark comprises 13,452 images, with 44,738 annotations describing 24,129 instances.

**Subject Labels.** We manually categorize each sentence within these expressions into one of the following subjects: appearance, human-object interaction (HOI), location, action, celebrity, optical character recognition (OCR), or None. The label criteria for each subject can be found in Section B.

**Data Format.** Each instance $I$ is associated with an image $X$, a bounding box $b = \{x, y, w, h\}$—where $(x, y)$ represents the coordinates of the top left corner, and $w$ and $h$ denote the width and height—and a referring expression $S = \{s_1, ..., s_n\}$ containing $N$ sentences. Each sentence $s_i$ within the expression $S$ has a subject label $l_i$.

## 3.2 Analysis

**Annotation Length.** Figure 3a visualizes the distribution of annotation lengths for four different benchmarks: HC-RefCOCO, HC-RefCOCO+, HC-RefCOCOg, and our HC-RefLoCo. The distribution of our HC-RefLoCo is markedly different from the other three benchmarks—there is a distinct peak around 100 words, indicating that the referring expressions in HC-RefLoCo are significantly longer than those in the others, which have peaks within the 4-8 word range. Additionally, the distribution of HC-RefLoCo spans from around 50 to 150 words, showing a much broader range.

**Sentence Length.** The HC-RefLoCo benchmark features annotations composed of multiple sentences. Figure 3b illustrates the distribution of sentence lengths across four benchmarks, using all sentences from all annotations for statistical analysis. Our benchmark notably peaks at a sentence length of

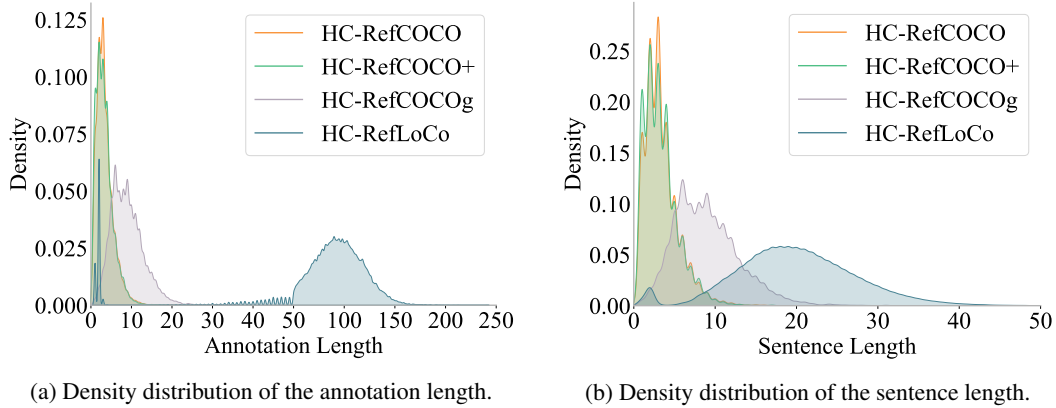

(a) Density distribution of the annotation length.

(b) Density distribution of the sentence length.

Figure 3: Statistical analysis of annotation length and sentence length across four benchmarks.

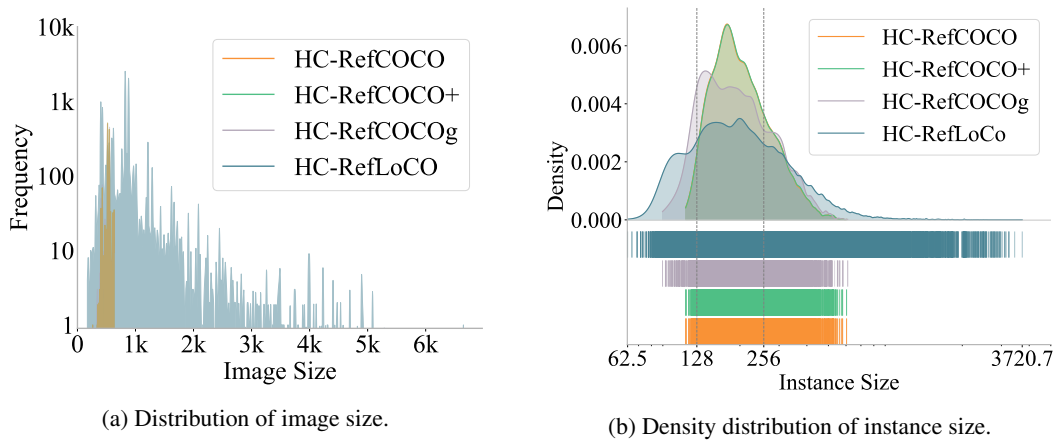

(a) Distribution of image size.

(b) Density distribution of instance size.

Figure 4: Statistical analysis of (a) the image size, and (b) the instance size, across four benchmarks. The instance size is represented by its square root. Note that there is a high distribution overlap among HC-RefCOCO, HC-RefCOCO+, and HC-RefCOCOg since they derive from the same dataset.

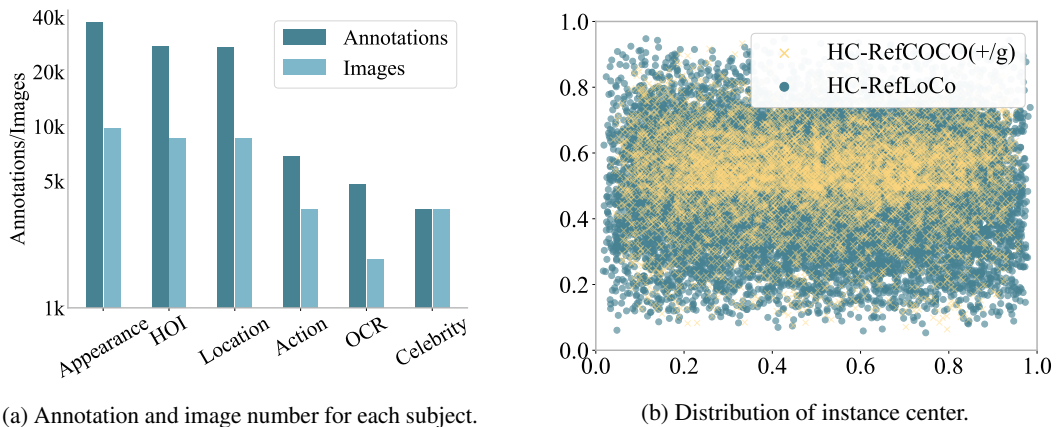

(a) Annotation and image number for each subject.

(b) Distribution of instance center.

Figure 5: (a) Per-subject analysis. (b) Distribution of instance center. We compare our HC-RefLoCo benchmark with the combination of HC-RefCOCO, HC-RefCOCO+ and HC-RefCOCOg.

approximately 18-20 words. In contrast, the other three benchmarks, HC-RefCOCO, HC-RefCOCO+, and HC-RefCOCOg, generally use single-sentence annotations, typically around 4-8 words.

Table 2: Performance evaluation across 24 models on our HC-RefLoCo benchmark. Models indicated with a † generate mask outputs, which we convert into tight bounding boxes to enable evaluation. Refer to Section D for the details of each model. NVIDIA A100 (80G) GPUs are used for evaluation.

| Model | Val+Test | | | | Val | Test |
|---|---|---|---|---|---|---|
| | $Acc_{0.5}$ | $Acc_{0.75}$ | $Acc_{0.9}$ | mAcc | mAcc | mAcc |
| GPT-4V [54–56] | 17.4 | 2.6 | 0.3 | 5.5 | 5.5 | 5.6 |
| GroundingGPT [36] | 56.6 | 27.2 | 5.3 | 29.8 | 30.0 | 29.8 |
| Ferret 7B [88] | 44.9 | 32.6 | 11.7 | 30.0 | 30.6 | 29.7 |
| Ferret 13B [88] | 52.9 | 38.5 | 15.6 | 35.7 | 35.9 | 35.6 |
| MiniGPT4-v2 [4] | 47.1 | 31.7 | 11.6 | 30.3 | 30.7 | 30.1 |
| KOSMOS-2 [59] | 45.3 | 38.0 | 20.0 | 34.1 | 34.2 | 34.0 |
| Shikra [5] | 56.8 | 35.6 | 10.3 | 34.4 | 34.6 | 34.3 |
| OFA [73] | 48.4 | 37.0 | 21.7 | 35.3 | 35.2 | 35.3 |
| OFA-Large[73] | 70.5 | 61.6 | 44.0 | 58.1 | 57.9 | 58.1 |
| Qwen-VL [3] | 67.9 | 56.8 | 34.8 | 52.8 | 53.1 | 52.6 |
| CogVLM [75] | 66.0 | 59.6 | 43.8 | 55.8 | 56.3 | 55.5 |
| Lenna [78] | 68.8 | 63.5 | 51.6 | 60.6 | 60.5 | 60.7 |
| ONE PEACE [74] | 79.3 | 69.0 | 43.8 | 63.1 | 63.4 | 62.9 |
| SPHINX-MoE [39] | 76.3 | 57.7 | 21.8 | 52.5 | 52.7 | 52.4 |
| SPHINX [39] | 77.5 | 61.0 | 27.0 | 55.4 | 55.8 | 55.2 |
| SPHINX-1k [39] | 80.7 | 68.6 | 41.1 | 63.0 | 63.0 | 62.9 |
| SPHINX-MoE-1k [39] | **85.8** | **77.3** | 53.7 | 71.4 | 71.5 | 71.4 |
| SPHINX-v2-1k [39] | 84.1 | 77.1 | **56.2** | **71.7** | 71.6 | **71.7** |
| PixelLM 7B† [98] | 38.5 | 24.7 | 11.8 | 24.5 | 24.6 | 24.4 |
| PixelLM 13B† [98] | 63.6 | 46.6 | 25.8 | 44.6 | 45.0 | 44.4 |
| LISA-explanatory† [30] | 47.6 | 37.6 | 27.0 | 36.7 | 36.7 | 36.7 |
| LISA† [30] | 52.4 | 42.1 | 31.3 | 41.1 | 41.1 | 41.1 |
| PSALM† [96] | 61.7 | 53.4 | 40.2 | 51.1 | 51.4 | 51.0 |
| GlaMM† [62] | **66.1** | **56.9** | **44.2** | **55.0** | 54.9 | **55.0** |

**Image Size.** Figure 4a compares the image size distributions of our benchmarks against HC-RefCOCO, HC-RefCOCO+, and HC-RefCOCOg. Since all three compared benchmarks derive from the same COCO dataset, there is a high distribution overlap. In contrast, our HC-RefLoCo benchmark covers a wider range of image sizes.

**Instance Size.** In Figure 4b, we visualize the instance size distribution across four benchmarks. Our benchmark spans a wider range of instance scales. The square root of the instance size varies from 62.5 to 3720.7, with an average of 313.8.

**Annotation and Image Number for Each Subject.** In our HC-RefLoCo benchmark, each annotation is a referring expression composed of multiple sentences for a given instance. Each sentence is assigned a specific subject label. As illustrated in Figure 5a, we analyze the number of annotations containing at least one sentence with the corresponding subject label for each subject.

**Instance Center.** Figure 5b presents a scatter plot illustrating the distribution of instance centers across two datasets: our HC-RefLoCo benchmark and the combined datasets of HC-RefCOCO, HC-RefCOCO+, and HC-RefCOCOg. Our benchmark demonstrates a more uniform spatial distribution.

## 4 Evaluation

**Benchmark Usage.** Modern REC models are often trained on extensive and diverse datasets. For example, the SPHINX [39] model leverages a mix of 16 unimodal and multimodal datasets, encompassing millions of training samples. Our HC-RefLoCo benchmark is designed to assess the capabilities of these advanced models without imposing any limitations on the sources of training data. The benchmark is divided into two subsets: a validation set, comprising 30% of the data with 4,000 images, 7,190 instances, and 13,360 annotations; and a test set, comprising 70% of the data

Table 3: Per-subject evaluation across 24 models on our HC-RefLoCo. We report mAcc for each set.

| Model | Appearance | HOI | Celebrity | OCR | Action | Location |
|---|---|---|---|---|---|---|
| GPT-4V [54–56] | 5.0 | 5.1 | 12.0 | 5.1 | 3.6 | 4.6 |
| GroundingGPT [36] | 27.3 | 27.5 | 61.4 | 25.8 | 21.3 | 23.0 |
| Ferret 7B [88] | 27.9 | 27.9 | 57.0 | 27.0 | 24.2 | 25.1 |
| Ferret 13B [88] | 33.9 | 34.4 | 58.5 | 33.5 | 28.8 | 30.9 |
| MiniGPT4-v2 [4] | 27.4 | 27.5 | 66.2 | 24.6 | 22.6 | 22.7 |
| KOSMOS-2 [59] | 31.5 | 32.9 | 65.8 | 31.5 | 27.9 | 28.2 |
| Shikra [5] | 32.7 | 32.5 | 55.9 | 29.7 | 30.6 | 31.7 |
| OFA [73] | 35.2 | 35.3 | 36.8 | 35.2 | 32.3 | 32.2 |
| OFA Large[73] | 58.4 | 58.3 | 56.0 | 56.9 | 55.1 | 55.2 |
| Qwen-VL [3] | 52.7 | 53.1 | 56.1 | 50.9 | 47.8 | 49.3 |
| CogVLM [75] | 54.8 | 53.6 | 66.9 | 50.3 | 55.9 | 55.2 |
| Lenna [78] | 61.8 | 62.3 | 50.6 | 61.6 | 56.5 | 57.2 |
| ONE PEACE [74] | 62.1 | 63.5 | **75.4** | 62.1 | 55.8 | 56.6 |
| SPHINX-MoE [39] | 51.6 | 52.9 | 64.4 | 52.1 | 45.5 | 47.9 |
| SPHINX [39] | 54.2 | 55.1 | 70.4 | 53.1 | 49.4 | 50.8 |
| SPHINX-1k [39] | 62.7 | 63.3 | 66.0 | 61.7 | 59.0 | 59.6 |
| SPHINX-MoE-1k [39] | 71.8 | 72.4 | 67.7 | 72.0 | 67.9 | 68.9 |
| SPHINX-v2-1k [39] | **72.4** | **73.0** | 64.1 | **72.3** | **68.7** | **69.6** |
| PixelLM 7B[†] [98] | 23.3 | 22.6 | 39.6 | 23.4 | 22.4 | 20.9 |
| PixelLM 13B[†] [98] | 43.8 | 44.9 | 54.8 | 44.0 | 38.9 | 40.3 |
| LISA-explanatory[†] [30] | 34.1 | 32.5 | 69.6 | 30.8 | 33.1 | 31.2 |
| LISA[†] [30] | 38.8 | 38.0 | **70.2** | 36.7 | 37.1 | 35.0 |
| PSALM[†] [96] | 51.7 | 51.6 | 47.3 | **52.2** | 48.3 | 49.5 |
| GlaMM[†] [62] | **54.0** | **53.4** | 68.7 | 51.7 | **51.3** | **51.3** |

with 9,452 images, 16,939 instances, and 31,378 annotations. While we provide these two splits, we encourage the combined use of both validation and test sets for model evaluation, particularly in the current era of large-scale multimodal models, where the use of unrestricted training data is common.

**Evaluation Protocols.** In the conventional evaluation protocol, an instance is deemed successfully located if the IoU between the predicted bounding box and the ground truth surpasses 0.5. Accuracy is then employed as the evaluation metric, known as $Acc_{0.5}$. To provide a more comprehensive assessment of model performance, we propose three evaluation protocols:

- In addition to $Acc_{0.5}$, we also measure $Acc_{0.75}$, $Acc_{0.9}$, and the mean accuracy (mAcc), which is the average of $Acc_{0.5}$ through $Acc_{0.95}$ at intervals of 0.05.

- Figure 5a presents the number of annotations for each subject. We further conduct a per-subject evaluation using mAcc as the evaluation metric.

- To assess robustness to variations in instance sizes, we report $mAcc_s$, $mAcc_m$, and $mAcc_l$, representing the mAcc for small, medium, and large instances. The size of an instance is determined by taking the square root of its area. Instances are categorized as small if their size is less than 128, medium if their size ranges from 128 to 256, and large if their size exceeds 256.

## 5 Experiments

**Main Results.** We assess a total of 24 advanced models, which are divided into two categories based on their output types: 1) models producing bounding box outputs, including GPT-4V [54–56], GroundingGPT [36], Ferret [88], MiniGPT4-v2 [101, 4], KOSMOS-2 [59], Shikra [5], OFA [73], Qwen-VL [3], CogVLM [75], Lenna [78], ONE-PEACE [74], and SPHINX [15, 39], and 2) models generating mask outputs, including PixelLM [98], LISA [30], PSALM [96], and GlaMM [62]. The specific prompt used for GPT-4V evaluation is described in Section A.4. For models that produce mask outputs, we convert these masks into tight bounding boxes to facilitate evaluation. The performance results are presented in Table 2.

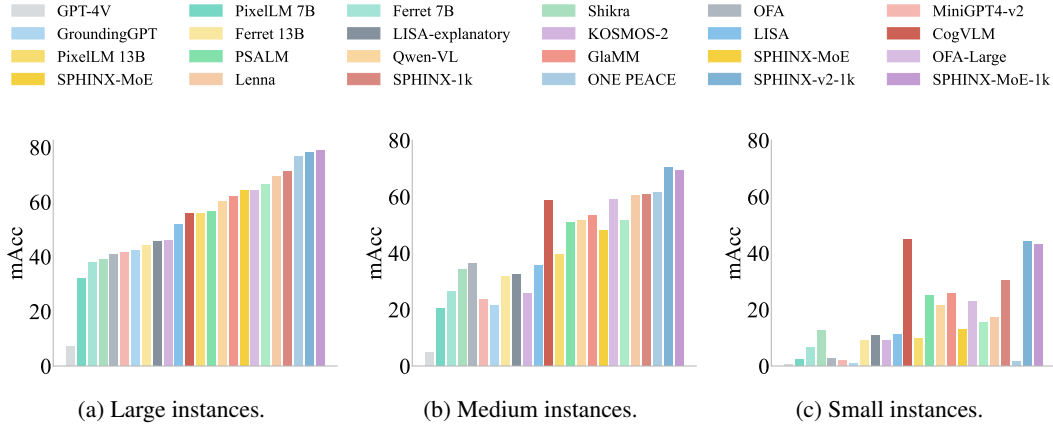

Figure 6: Scale-aware evaluation. Models are sorted in ascending order based on their performance on large instances. We use mAcc as the evaluation metric.

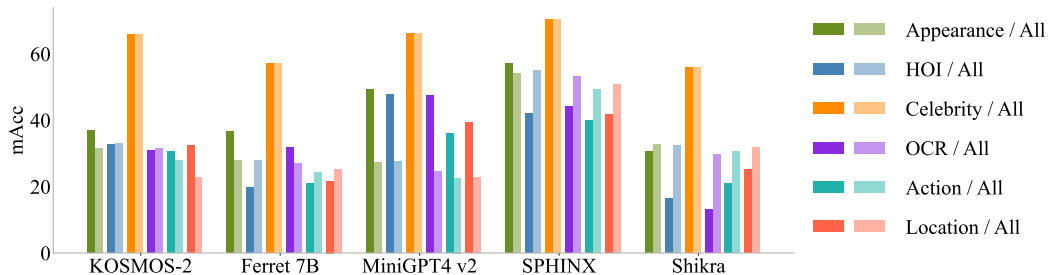

Figure 7: Per-subject evaluation under two scenarios: 1) using the original annotations (denoted as "All"); 2) retaining only sentences that correspond to the specific subject while discarding the rest for each annotation.

**Per-Subject Evaluation.** As outlined in Section 3, our benchmark is divided into six subsets, each corresponding to one of the following subjects: appearance, human-object interaction (HOI), location, action, celebrity, and optical character recognition (OCR). This segmentation enables a focused evaluation of the model's performance on specific topics. Table 3 presents the mAcc for each subset. The SPHINX-v2-1k [39] model demonstrates the highest overall performance across most subsets, while ONE-PEACE [74] excels particularly in celebrity recognition.

**Scale-Aware Evaluation.** In Figure 6, we assess model performance across three groups categorized by instance size: large, medium, and small. The size of each instance is determined by the square root of its area. Specifically, small instances have a size less than 128, medium instances range from 128 to 256, and large instances exceed 256. Generally, most models exhibit a decline in performance as instance size decreases. CogVLM [75] shows the greatest robustness across different instance scales.

**Effects of Using Detailed and Contextual Annotations.** In our HC-RefLoCo benchmark, each annotation comprises multiple sentences, with each sentence labeled to a specific subject. We conduct per-subject evaluations under two scenarios: 1) using the original annotations; and 2) retaining only sentences that correspond to the specific subject while discarding the rest for each annotation. In Figure 7, we assess five models—KOSMOS-2 [59], Ferret 7B [88], MiniGPT4 v2 [4], SPHINX [39], and Shikra [5]—each employing different language encoders: KOSMOS 1.3B [20], Vicuna 7B [8], LLaMa2 Chat 7B [72], LLaMa 2 13B [72], and LLaMa 7B [71], respectively. For most subjects, SPHINX and Shikra achieve higher performance when all sentences are used to describe the target instance, possibly due to their strong language encoders, LLaMa 2 13B [72] and LLaMa 7B [71]. Conversely, we observe that MiniGPT4 v2 [4] shows a significant decline in performance with annotations containing more contextual descriptions, highlighting its limitations in associating lengthy text descriptions with visual elements.

Additionally, we generate three extra sets by randomly selecting 1, 3, and 5 sentences from each annotation. In Figure 8 of Section C, we evaluate the same models on these three sets, alongside the original HC-RefLoCo benchmark. The results reveal that different models perform optimally on different sets, highlighting a trade-off—while longer descriptions offer more context, the models may fail to effectively associate the extended context with the target instance.

# 6 Conclusion

This paper introduces a novel benchmark called HC-RefLoCo, designed specifically for human-centric referring expression comprehension. HC-RefLoCo presents unique challenges and evaluation criteria for large multimodal models. Key features include: 1) a substantial benchmark with 13,452 images, 24,129 instances, and 44,738 annotations; 2) detailed annotations ranging from 15 to 241 words, with an average of 93.2 words, and an extensive vocabulary of 18,681 words; 3) sentence-level subject labels; 4) a wide range of instance scales; and 5) multiple evaluation protocols, including the utilization of various IoU criteria, subject-specific evaluations, and scale-aware evaluations. The benchmark and evaluation code will be publicly available to support the advancement of REC models, particularly in the LMM era.

**Acknowledgements.** This work was supported in part by the Australian Research Council under Projects DP240101848 and FT230100549.

## Footnotes

[2]https://github.com/wildchlamydia/mivolo

# References

[1] J.-B. Alayrac, J. Donahue, P. Luc, A. Miech, I. Barr, Y. Hasson, K. Lenc, A. Mensch, K. Millican, M. Reynolds, et al. Flamingo: a visual language model for few-shot learning. *Advances in neural information processing systems*, 35:23716–23736, 2022.

[2] A. Awadalla, I. Gao, J. Gardner, J. Hessel, Y. Hanafy, W. Zhu, K. Marathe, Y. Bitton, S. Gadre, S. Sagawa, et al. OpenFlamingo: An open-source framework for training large autoregressive vision-language models. *arXiv preprint arXiv:2308.01390*, 2023.

[3] J. Bai, S. Bai, S. Yang, S. Wang, S. Tan, P. Wang, J. Lin, C. Zhou, and J. Zhou. Qwen-VL: A versatile vision-language model for understanding, localization, text reading, and beyond. *arXiv preprint arXiv:2308.12966*, 2023.

[4] J. Chen, D. Zhu, X. Shen, X. Li, Z. Liu, P. Zhang, R. Krishnamoorthi, V. Chandra, Y. Xiong, and M. Elhoseiny. MiniGPT-v2: large language model as a unified interface for vision-language multi-task learning. *arXiv preprint arXiv:2310.09478*, 2023.

[5] K. Chen, Z. Zhang, W. Zeng, R. Zhang, F. Zhu, and R. Zhao. Shikra: Unleashing multimodal LLM's referential dialogue magic. *arXiv preprint arXiv:2306.15195*, 2023.

[6] L. Chen, J. Li, X. Dong, P. Zhang, C. He, J. Wang, F. Zhao, and D. Lin. ShareGPT4V: Improving large multi-modal models with better captions. *arXiv preprint arXiv:2311.12793*, 2023.

[7] B. Cheng, I. Misra, A. G. Schwing, A. Kirillov, and R. Girdhar. Masked-attention mask transformer for universal image segmentation. 2022.

[8] W.-L. Chiang, Z. Li, Z. Lin, Y. Sheng, Z. Wu, H. Zhang, L. Zheng, S. Zhuang, Y. Zhuang, J. E. Gonzalez, I. Stoica, and E. P. Xing. Vicuna: An open-source chatbot impressing GPT-4 with 90%* chatgpt quality, March 2023.

[9] X. Chu, A. Zheng, X. Zhang, and J. Sun. Detection in crowded scenes: One proposal, multiple predictions. In *Proceedings of the IEEE/CVF conference on computer vision and pattern recognition*, pages 12214–12223, 2020.

[10] J. Deng, J. Guo, Y. Zhou, J. Yu, I. Kotsia, and S. Zafeiriou. RetinaFace: Single-stage dense face localisation in the wild. *arXiv preprint arXiv:1905.00641*, 2019.

[11] Y. Du, F. Wei, Z. Zhang, M. Shi, Y. Gao, and G. Li. Learning to prompt for open-vocabulary object detection with vision-language model. In *Proceedings of the IEEE/CVF Conference on Computer Vision and Pattern Recognition*, pages 14084–14093, 2022.

[12] C. Eom and B. Ham. Learning disentangled representation for robust person re-identification. *Advances in neural information processing systems*, 32, 2019.

[13] Y. Fang, W. Wang, B. Xie, Q. Sun, L. Wu, X. Wang, T. Huang, X. Wang, and Y. Cao. Eva: Exploring the limits of masked visual representation learning at scale. In *Proceedings of the IEEE/CVF Conference on Computer Vision and Pattern Recognition*, pages 19358–19369, 2023.

[14] C. Feichtenhofer, H. Fan, J. Malik, and K. He. SlowFast networks for video recognition. In *Proceedings of the IEEE/CVF international conference on computer vision*, pages 6202–6211, 2019.

[15] P. Gao, R. Zhang, C. Liu, L. Qiu, S. Huang, W. Lin, S. Zhao, S. Geng, Z. Lin, P. Jin, et al. SPHINX-X: Scaling data and parameters for a family of multi-modal large language models. *arXiv preprint arXiv:2402.05935*, 2024.

[16] X. Gu, T.-Y. Lin, W. Kuo, and Y. Cui. Open-vocabulary object detection via vision and language knowledge distillation. *arXiv preprint arXiv:2104.13921*, 2021.

[17] Q. Guo, S. De Mello, H. Yin, W. Byeon, K. C. Cheung, Y. Yu, P. Luo, and S. Liu. RegionGPT: Towards region understanding vision language model. *arXiv preprint arXiv:2403.02330*, 2024.

[18] K. He, X. Zhang, S. Ren, and J. Sun. Deep residual learning for image recognition. In *Proceedings of the IEEE conference on computer vision and pattern recognition*, pages 770–778, 2016.

[19] S. He, H. Luo, P. Wang, F. Wang, H. Li, and W. Jiang. TransReID: Transformer-based object re-identification. In *Proceedings of the IEEE/CVF international conference on computer vision*, pages 15013–15022, 2021.

[20] S. Huang, L. Dong, W. Wang, Y. Hao, S. Singhal, S. Ma, T. Lv, L. Cui, O. K. Mohammed, B. Patra, et al. Language is not all you need: Aligning perception with language models. *Advances in Neural Information Processing Systems*, 36, 2024.

[21] A. Q. Jiang, A. Sablayrolles, A. Roux, A. Mensch, B. Savary, C. Bamford, D. S. Chaplot, D. d. l. Casas, E. B. Hanna, F. Bressand, et al. Mixtral of experts. *arXiv preprint arXiv:2401.04088*, 2024.

[22] L. Jin, G. Luo, Y. Zhou, X. Sun, G. Jiang, A. Shu, and R. Ji. RefCLIP: A universal teacher for weakly supervised referring expression comprehension. In *Proceedings of the IEEE/CVF conference on computer vision and pattern recognition*, pages 2681–2690, 2023.

[23] S. Kazemzadeh, V. Ordonez, M. Matten, and T. Berg. ReferItGame: Referring to objects in photographs of natural scenes. In *Proceedings of the 2014 conference on empirical methods in natural language processing (EMNLP)*, pages 787–798, 2014.

[24] M. Kim, A. K. Jain, and X. Liu. AdaFace: Quality adaptive margin for face recognition. In *Proceedings of the IEEE/CVF conference on computer vision and pattern recognition*, pages 18750–18759, 2022.

[25] Y. Kim, W. Park, M.-C. Roh, and J. Shin. GroupFace: Learning latent groups and constructing group-based representations for face recognition. In *Proceedings of the IEEE/CVF Conference on Computer Vision and Pattern Recognition*, pages 5621–5630, 2020.

[26] A. Kirillov, E. Mintun, N. Ravi, H. Mao, C. Rolland, L. Gustafson, T. Xiao, S. Whitehead, A. C. Berg, W.-Y. Lo, P. Dollár, and R. Girshick. Segment anything. *arXiv:2304.02643*, 2023.

[27] A. Kirillov, E. Mintun, N. Ravi, H. Mao, C. Rolland, L. Gustafson, T. Xiao, S. Whitehead, A. C. Berg, W.-Y. Lo, et al. Segment anything. In *Proceedings of the IEEE/CVF International Conference on Computer Vision*, pages 4015–4026, 2023.

[28] I. Krasin, T. Duerig, N. Alldrin, V. Ferrari, S. Abu-El-Haija, A. Kuznetsova, H. Rom, J. Uijlings, S. Popov, A. Veit, et al. OpenImages: A public dataset for large-scale multi-label and multi-class image classification. *Dataset available from https://github. com/openimages*, 2(3):18, 2017.

[29] R. Krishna, Y. Zhu, O. Groth, J. Johnson, K. Hata, J. Kravitz, S. Chen, Y. Kalantidis, L.-J. Li, D. A. Shamma, et al. Visual Genome: Connecting language and vision using crowdsourced dense image annotations. *International journal of computer vision*, 123:32–73, 2017.

[30] X. Lai, Z. Tian, Y. Chen, Y. Li, Y. Yuan, S. Liu, and J. Jia. LISA: Reasoning segmentation via large language model. *arXiv preprint arXiv:2308.00692*, 2023.

[31] M. Lewis, Y. Liu, N. Goyal, M. Ghazvininejad, A. Mohamed, O. Levy, V. Stoyanov, and L. Zettlemoyer. BART: denoising sequence-to-sequence pre-training for natural language generation, translation, and comprehension. *CoRR*, abs/1910.13461, 2019. URL http://arxiv.org/abs/1910.13461.

[32] J. Li, Y. Wang, C. Wang, Y. Tai, J. Qian, J. Yang, C. Wang, J. Li, and F. Huang. DSFD: dual shot face detector. In *Proceedings of the IEEE/CVF conference on computer vision and pattern recognition*, pages 5060–5069, 2019.

[33] J. Li, D. Li, S. Savarese, and S. Hoi. BLIP-2: Bootstrapping language-image pre-training with frozen image encoders and large language models. In *International conference on machine learning*, pages 19730–19742. PMLR, 2023.

[34] Y. Li, B. Ji, X. Shi, J. Zhang, B. Kang, and L. Wang. TEA: Temporal excitation and aggregation for action recognition. In *Proceedings of the IEEE/CVF conference on computer vision and pattern recognition*, pages 909–918, 2020.

[35] Y. Li, S. Bubeck, R. Eldan, A. Del Giorno, S. Gunasekar, and Y. T. Lee. Textbooks are all you need ii: phi-1.5 technical report. *arXiv preprint arXiv:2309.05463*, 2023.

[36] Z. Li, Q. Xu, D. Zhang, H. Song, Y. Cai, Q. Qi, R. Zhou, J. Pan, Z. Li, V. T. Vu, et al. LEGO: Language enhanced multi-modal grounding model. *arXiv preprint arXiv:2401.06071*, 2024.

[37] Y. Liao, S. Liu, G. Li, F. Wang, Y. Chen, C. Qian, and B. Li. A real-time cross-modality correlation filtering method for referring expression comprehension. In *Proceedings of the IEEE/CVF Conference on Computer Vision and Pattern Recognition*, pages 10880–10889, 2020.

[38] T.-Y. Lin, M. Maire, S. Belongie, J. Hays, P. Perona, D. Ramanan, P. Dollár, and C. L. Zitnick. Microsoft COCO: Common objects in context. In *Computer Vision–ECCV 2014: 13th European Conference, Zurich, Switzerland, September 6-12, 2014, Proceedings, Part V 13*, pages 740–755. Springer, 2014.

[39] Z. Lin, C. Liu, R. Zhang, P. Gao, L. Qiu, H. Xiao, H. Qiu, C. Lin, W. Shao, K. Chen, et al. SPHINX: The joint mixing of weights, tasks, and visual embeddings for multi-modal large language models. *arXiv preprint arXiv:2311.07575*, 2023.

[40] D. Liu, H. Zhang, F. Wu, and Z.-J. Zha. Learning to assemble neural module tree networks for visual grounding. In *Proceedings of the IEEE/CVF International Conference on Computer Vision*, pages 4673–4682, 2019.

[41] H. Liu, C. Li, Q. Wu, and Y. J. Lee. Visual instruction tuning, 2023.

[42] J. Liu, Z.-J. Zha, D. Chen, R. Hong, and M. Wang. Adaptive transfer network for cross-domain person re-identification. In *Proceedings of the IEEE/CVF conference on computer vision and pattern recognition*, pages 7202–7211, 2019.

[43] S. Liu, Z. Zeng, T. Ren, F. Li, H. Zhang, J. Yang, C. Li, J. Yang, H. Su, J. Zhu, et al. Grounding DINO: Marrying DINO with grounded pre-training for open-set object detection. *arXiv preprint arXiv:2303.05499*, 2023.

[44] X. Liu, Z. Wang, J. Shao, X. Wang, and H. Li. Improving referring expression grounding with cross-modal attention-guided erasing. In *Proceedings of the IEEE/CVF conference on computer vision and pattern recognition*, pages 1950–1959, 2019.

[45] Z. Liu, H. Zhang, Z. Chen, Z. Wang, and W. Ouyang. Disentangling and unifying graph convolutions for skeleton-based action recognition. In *Proceedings of the IEEE/CVF conference on computer vision and pattern recognition*, pages 143–152, 2020.

[46] Z. Liu, Y. Lin, Y. Cao, H. Hu, Y. Wei, Z. Zhang, S. Lin, and B. Guo. Swin transformer: Hierarchical vision transformer using shifted windows. In *Proceedings of the IEEE/CVF International Conference on Computer Vision (ICCV)*, 2021.

[47] G. Luo, Y. Zhou, R. Ji, X. Sun, J. Su, C.-W. Lin, and Q. Tian. Cascade grouped attention network for referring expression segmentation. In *Proceedings of the 28th ACM International Conference on Multimedia*, pages 1274–1282, 2020.

[48] G. Luo, Y. Zhou, X. Sun, L. Cao, C. Wu, C. Deng, and R. Ji. Multi-task collaborative network for joint referring expression comprehension and segmentation. In *Proceedings of the IEEE/CVF Conference on computer vision and pattern recognition*, pages 10034–10043, 2020.

[49] H. Luo, Y. Gu, X. Liao, S. Lai, and W. Jiang. Bag of tricks and a strong baseline for deep person re-identification. In *Proceedings of the IEEE/CVF conference on computer vision and pattern recognition workshops*, 2019.

[50] J. Mao, J. Huang, A. Toshev, O. Camburu, A. L. Yuille, and K. Murphy. Generation and comprehension of unambiguous object descriptions. In *Proceedings of the IEEE conference on computer vision and pattern recognition*, pages 11–20, 2016.

[51] V. Mazzia, S. Angarano, F. Salvetti, F. Angelini, and M. Chiaberge. Action transformer: A self-attention model for short-time pose-based human action recognition. *Pattern Recognition*, 124:108487, 2022.

[52] Q. Meng, S. Zhao, Z. Huang, and F. Zhou. MagFace: A universal representation for face recognition and quality assessment. In *Proceedings of the IEEE/CVF conference on computer vision and pattern recognition*, pages 14225–14234, 2021.

[53] X. Ming, F. Wei, T. Zhang, D. Chen, and F. Wen. Group sampling for scale invariant face detection. In *Proceedings of the IEEE/CVF Conference on Computer Vision and Pattern Recognition*, pages 3446–3456, 2019.

[54] OpenAI. Gpt-4 technical report, 2023.

[55] OpenAI. Gpt-4v(ision) system card. 2023. URL https://cdn.openai.com/papers/GPTV_System_Card.pdf.

[56] OpenAI. Gpt-4v(ision) technical work and authors. 2023. URL https://cdn.openai.com/contributions/gpt-4v.pdf.

[57] M. Oquab, T. Darcet, T. Moutakanni, H. V. Vo, M. Szafraniec, V. Khalidov, P. Fernandez, D. Haziza, F. Massa, A. El-Nouby, R. Howes, P.-Y. Huang, H. Xu, V. Sharma, S.-W. Li, W. Galuba, M. Rabbat, M. Assran, N. Ballas, G. Synnaeve, I. Misra, H. Jegou, J. Mairal, P. Labatut, A. Joulin, and P. Bojanowski. Dinov2: Learning robust visual features without supervision, 2023.

[58] K. Park, T. Patten, and M. Vincze. Pix2Pose: Pixel-wise coordinate regression of objects for 6d pose estimation. In *Proceedings of the IEEE/CVF International Conference on Computer Vision*, pages 7668–7677, 2019.

[59] Z. Peng, W. Wang, L. Dong, Y. Hao, S. Huang, S. Ma, and F. Wei. Kosmos-2: Grounding multimodal large language models to the world. *arXiv preprint arXiv:2306.14824*, 2023.

[60] B. A. Plummer, L. Wang, C. M. Cervantes, J. C. Caicedo, J. Hockenmaier, and S. Lazebnik. Flickr30k entities: Collecting region-to-phrase correspondences for richer image-to-sentence models. *IJCV*, 123(1): 74–93, 2017.

[61] A. Radford, J. W. Kim, C. Hallacy, A. Ramesh, G. Goh, S. Agarwal, G. Sastry, A. Askell, P. Mishkin, J. Clark, et al. Learning transferable visual models from natural language supervision. In *International conference on machine learning*, pages 8748–8763. PMLR, 2021.

[62] H. Rasheed, M. Maaz, S. Shaji, A. Shaker, S. Khan, H. Cholakkal, R. M. Anwer, E. Xing, M.-H. Yang, and F. S. Khan. GLaMM: Pixel grounding large multimodal model. *arXiv preprint arXiv:2311.03356*, 2023.

[63] F. Schroff, D. Kalenichenko, and J. Philbin. Facenet: A unified embedding for face recognition and clustering. In *Proceedings of the IEEE conference on computer vision and pattern recognition*, pages 815–823, 2015.

[64] C. Schuhmann, R. Beaumont, R. Vencu, C. W. Gordon, R. Wightman, M. Cherti, T. Coombes, A. Katta, C. Mullis, M. Wortsman, P. Schramowski, S. R. Kundurthy, K. Crowson, L. Schmidt, R. Kaczmarczyk, and J. Jitsev. LAION-5b: An open large-scale dataset for training next generation image-text models. In *Thirty-sixth Conference on Neural Information Processing Systems Datasets and Benchmarks Track*, 2022. URL https://openreview.net/forum?id=M3Y74vmsMcY.

[65] S. Shao, Z. Li, T. Zhang, C. Peng, G. Yu, X. Zhang, J. Li, and J. Sun. Objects365: A large-scale, high-quality dataset for object detection. In *Proceedings of the IEEE/CVF international conference on computer vision*, pages 8430–8439, 2019.

[66] K. Sun, B. Xiao, D. Liu, and J. Wang. Deep high-resolution representation learning for human pose estimation. In *Proceedings of the IEEE/CVF conference on computer vision and pattern recognition*, pages 5693–5703, 2019.

[67] M. Sun, W. Suo, P. Wang, Y. Zhang, and Q. Wu. A proposal-free one-stage framework for referring expression comprehension and generation via dense cross-attention. *IEEE Transactions on Multimedia*, 2022.

[68] Y. Taigman, M. Yang, M. Ranzato, and L. Wolf. Deepface: Closing the gap to human-level performance in face verification. In *Proceedings of the IEEE conference on computer vision and pattern recognition*, pages 1701–1708, 2014.

[69] X. Tang, D. K. Du, Z. He, and J. Liu. Pyramidbox: A context-assisted single shot face detector. In *Proceedings of the European conference on computer vision (ECCV)*, pages 797–813, 2018.

[70] G. Team, R. Anil, S. Borgeaud, Y. Wu, J.-B. Alayrac, J. Yu, R. Soricut, J. Schalkwyk, A. M. Dai, A. Hauth, et al. Gemini: a family of highly capable multimodal models. *arXiv preprint arXiv:2312.11805*, 2023.

[71] H. Touvron, T. Lavril, G. Izacard, X. Martinet, M.-A. Lachaux, T. Lacroix, B. Rozière, N. Goyal, E. Hambro, F. Azhar, et al. Llama: Open and efficient foundation language models. *arXiv preprint arXiv:2302.13971*, 2023.

[72] H. Touvron, L. Martin, K. Stone, P. Albert, A. Almahairi, Y. Babaei, N. Bashlykov, S. Batra, P. Bhargava, S. Bhosale, et al. Llama 2: Open foundation and fine-tuned chat models. *arXiv preprint arXiv:2307.09288*, 2023.

[73] P. Wang, A. Yang, R. Men, J. Lin, S. Bai, Z. Li, J. Ma, C. Zhou, J. Zhou, and H. Yang. Ofa: Unifying architectures, tasks, and modalities through a simple sequence-to-sequence learning framework. In *International Conference on Machine Learning*, pages 23318–23340. PMLR, 2022.

[74] P. Wang, S. Wang, J. Lin, S. Bai, X. Zhou, J. Zhou, X. Wang, and C. Zhou. One-peace: Exploring one general representation model toward unlimited modalities. *arXiv preprint arXiv:2305.11172*, 2023.

[75] W. Wang, Q. Lv, W. Yu, W. Hong, J. Qi, Y. Wang, J. Ji, Z. Yang, L. Zhao, X. Song, et al. Cogvlm: Visual expert for pretrained language models. *arXiv preprint arXiv:2311.03079*, 2023.

[76] X. Wang, T. Xiao, Y. Jiang, S. Shao, J. Sun, and C. Shen. Repulsion loss: Detecting pedestrians in a crowd. In *Proceedings of the IEEE conference on computer vision and pattern recognition*, pages 7774–7783, 2018.

[77] F. Wei, X. Sun, H. Li, J. Wang, and S. Lin. Point-set anchors for object detection, instance segmentation and pose estimation. In *Computer Vision–ECCV 2020: 16th European Conference, Glasgow, UK, August 23–28, 2020, Proceedings, Part X 16*, pages 527–544. Springer, 2020.

[78] F. Wei, X. Zhang, A. Zhang, B. Zhang, and X. Chu. Lenna: Language enhanced reasoning detection assistant. *arXiv preprint arXiv:2312.02433*, 2023.

[79] M. Xu, Z. Zhang, F. Wei, Y. Lin, Y. Cao, H. Hu, and X. Bai. A simple baseline for open-vocabulary semantic segmentation with pre-trained vision-language model. In *European Conference on Computer Vision*, pages 736–753. Springer, 2022.

[80] M. Xu, Z. Zhang, F. Wei, H. Hu, and X. Bai. Side adapter network for open-vocabulary semantic segmentation. In *Proceedings of the IEEE/CVF Conference on Computer Vision and Pattern Recognition*, pages 2945–2954, 2023.

[81] Y. Xu, W. Yan, G. Yang, J. Luo, T. Li, and J. He. CenterFace: joint face detection and alignment using face as point. *Scientific Programming*, 2020:1–8, 2020.

[82] Y. Xu, F. Wei, X. Sun, C. Yang, Y. Shen, B. Dai, B. Zhou, and S. Lin. Cross-model pseudo-labeling for semi-supervised action recognition. In *Proceedings of the IEEE/CVF Conference on Computer Vision and Pattern Recognition*, pages 2959–2968, 2022.

[83] S. Yang, G. Li, and Y. Yu. Dynamic graph attention for referring expression comprehension. In *Proceedings of the IEEE/CVF International Conference on Computer Vision*, pages 4644–4653, 2019.

[84] S. Yang, T. Qu, X. Lai, Z. Tian, B. Peng, S. Liu, and J. Jia. An improved baseline for reasoning segmentation with large language model. *arXiv preprint arXiv:2312.17240*, 2023.

[85] Z. Yang, L. Li, K. Lin, J. Wang, C.-C. Lin, Z. Liu, and L. Wang. The dawn of lmms: Preliminary explorations with gpt-4v (ision). *arXiv preprint arXiv:2309.17421*, 9(1):1, 2023.

[86] Z. Yang, M. Lin, X. Zhong, Y. Wu, and Z. Wang. Good is bad: Causality inspired cloth-debiasing for cloth-changing person re-identification. In *Proceedings of the IEEE/CVF Conference on Computer Vision and Pattern Recognition*, pages 1472–1481, 2023.

[87] Q. Ye, H. Xu, G. Xu, J. Ye, M. Yan, Y. Zhou, J. Wang, A. Hu, P. Shi, Y. Shi, et al. mPLUG-Owl: Modularization empowers large language models with multimodality. *arXiv preprint arXiv:2304.14178*, 2023.

[88] H. You, H. Zhang, Z. Gan, X. Du, B. Zhang, Z. Wang, L. Cao, S.-F. Chang, and Y. Yang. Ferret: Refer and ground anything anywhere at any granularity. *arXiv preprint arXiv:2310.07704*, 2023.

[89] P. Young, A. Lai, M. Hodosh, and J. Hockenmaier. From image descriptions to visual denotations: New similarity metrics for semantic inference over event descriptions. *TACL*, 2:67–78, 2014.

[90] L. Yu, Z. Lin, X. Shen, J. Yang, X. Lu, M. Bansal, and T. L. Berg. MAttNet: Modular attention network for referring expression comprehension. In *Proceedings of the IEEE conference on computer vision and pattern recognition*, pages 1307–1315, 2018.

[91] F. Zhang, X. Zhu, H. Dai, M. Ye, and C. Zhu. Distribution-aware coordinate representation for human pose estimation. In *Proceedings of the IEEE/CVF conference on computer vision and pattern recognition*, pages 7093–7102, 2020.

[92] Q. Zhang, J. Zhang, Y. Xu, and D. Tao. Vision transformer with quadrangle attention. *arXiv preprint arXiv:2303.15105*, 2023.

[93] S. Zhang, X. Zhu, Z. Lei, H. Shi, X. Wang, and S. Z. Li. Faceboxes: A CPU real-time face detector with high accuracy. In *2017 IEEE International Joint Conference on Biometrics (IJCB)*, pages 1–9. IEEE, 2017.

[94] S. Zhang, X. Zhu, Z. Lei, H. Shi, X. Wang, and S. Z. Li. S3FD: Single shot scale-invariant face detector. In *Proceedings of the IEEE international conference on computer vision*, pages 192–201, 2017.

[95] S. Zhang, L. Wen, X. Bian, Z. Lei, and S. Z. Li. Occlusion-aware R-CNN: Detecting pedestrians in a crowd. In *Proceedings of the European conference on computer vision (ECCV)*, pages 637–653, 2018.

[96] Z. Zhang, Y. Ma, E. Zhang, and X. Bai. Psalm: Pixelwise segmentation with large multi-modal model, 2024.

[97] L. Zheng, V. Noroozi, and P. S. Yu. Joint deep modeling of users and items using reviews for recommendation. In *Proceedings of the tenth ACM international conference on web search and data mining*, pages 425–434, 2017.

[98] Y. W. Y. Z. D. F. J. F. X. J. Zhongwei Ren, Zhicheng Huang. PixelLM: Pixel reasoning with large multimodal model. *arXiv preprint arXiv:2312.02228*, 2023.

[99] Y. Zhou, R. Ji, G. Luo, X. Sun, J. Su, X. Ding, C.-W. Lin, and Q. Tian. A real-time global inference network for one-stage referring expression comprehension. *IEEE Transactions on Neural Networks and Learning Systems*, 34(1):134–143, 2021.

[100] C. Zhu, Y. Zhou, Y. Shen, G. Luo, X. Pan, M. Lin, C. Chen, L. Cao, X. Sun, and R. Ji. SeqTR: A simple yet universal network for visual grounding. In *European Conference on Computer Vision*, pages 598–615. Springer, 2022.

[101] D. Zhu, J. Chen, X. Shen, X. Li, and M. Elhoseiny. MiniGPT-4: Enhancing vision-language understanding with advanced large language models. *arXiv preprint arXiv:2304.10592*, 2023.

# A Prompts

## A.1 Prompt for Instance Description Generation

---

You are an advanced referring expression generator tasked with crafting a detailed and precise description of a person in an image. To achieve this, please adhere to the following guidelines:

1. Highlight unique characteristics that make the person distinctive.

2. Provide a comprehensive description of the person's overall appearance.

3. Mention any interactions the person has with objects or other people.

4. Include any visible text on the individual, such as text on clothing.

5. Detail any specific activities the person is engaged in.

6. Describe the person's location within the scene.

7. When multiple individuals have similar appearances, use their relative positions for identification, such as "the first person on the left" or "the individual in the middle of the second row".

Input image: *<Cropped Image>*.

---

## A.2 Prompt for Contextual Description Generation

---

You are an advanced referring expression generator tasked with crafting a detailed and precise description of a person highlighted by a red circle in an image. An initial description is provided as a reference. The description is *<Instance-Level Description>*. To achieve this, please adhere to the following guidelines:

1. Highlight unique characteristics that make the person distinctive.

2. Provide a comprehensive description of the person's overall appearance.

3. Mention any interactions the person has with objects or other people.

4. Include any visible text on the individual, such as text on clothing.

5. Detail any specific activities the person is engaged in.

6. Describe the person's location within the scene.

7. When multiple individuals have similar appearances, use their relative positions for identification, such as "the first person on the left" or "the individual in the middle of the second row".

Input image: [*Raw Image*].

---

## A.3 Prompt for Annotation Expansion

---

The following paragraph should be rewritten while retaining the essential information. Different expressions should be used, and the paragraph may be reorganized if necessary. The paragraph should not be altered merely by converting the passive voice to active voice or vice versa.

---

### A.4 Prompt for GPT-4V Evaluation

---

Given an image and a referring expression describing an instance visible in the image, the task is to identify the specific instance and output a bounding box in the format $(x, y, h, w)$, where $(x, y)$ represents the top-left corner and $(h, w)$ denotes the height and width. Ensure the response includes only the coordinates as described, without any additional text, characters, or spaces.

Input image: [*Raw Image*]

Description: [*Referring Expression of a Target Instance*]

---

## B  Labeling Criteria for Sentence-Level Annotations

As outlined in Section 3.1, each referring expression annotation in our HC-RefLoCo benchmark consists of multiple sentences. Each sentence is manually categorized into one of the following subjects: appearance, human-object interaction (HOI), location, action, celebrity, and optical character recognition (OCR). The specific labeling criteria for each subject are as follows:

- *Appearance.* Descriptions that focus on the physical attributes or visual characteristics of humans.
- *HOI.* Descriptions that detail the interactions between a person and objects.
- *Location.* Descriptions that specify the setting or place where the person is situated.
- *Action.* Descriptions that highlight the activities or movements of the person.
- *Celebrity.* Descriptions that identify the person as a famous individual or a well-known personality.
- *OCR.* Descriptions that mention text associated with the person, which can be read or recognized.

## C  More Analysis

**Using Randomly Selected Sentences as Referring Expressions.** We create three additional sets by randomly selecting 1, 3 and 5 sentences from each annotation. In Figure 8 we report the performance of five models on these three sets.

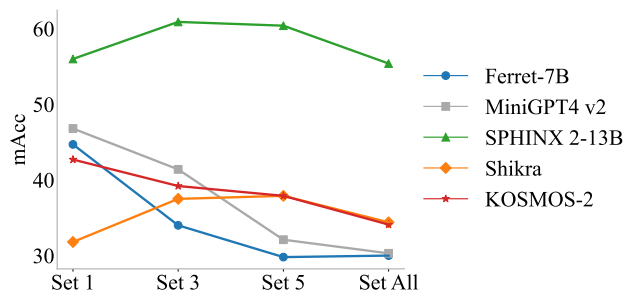

Figure 8: Alongside the original benchmark, we create three additional sets by randomly selecting 1, 3 and 5 sentences from each annotation. These sets are referred to as "Set-1," "Set-3," and "Set-5," respectively. We report mAcc on the four sets across five models.

**Statistics of Validation and Test Sets.** In Section 4, our benchmark is partitioned into a validation set and a test set. Figure 9 illustrates the number of annotations and images for each subject in both the validation set and the test set.

**Word Frequency.** Figure 10 illustrates the 20 most frequently used nouns in annotations across four different benchmarks. In our benchmark, the top 20 nouns are "person", "shirt", "hair",

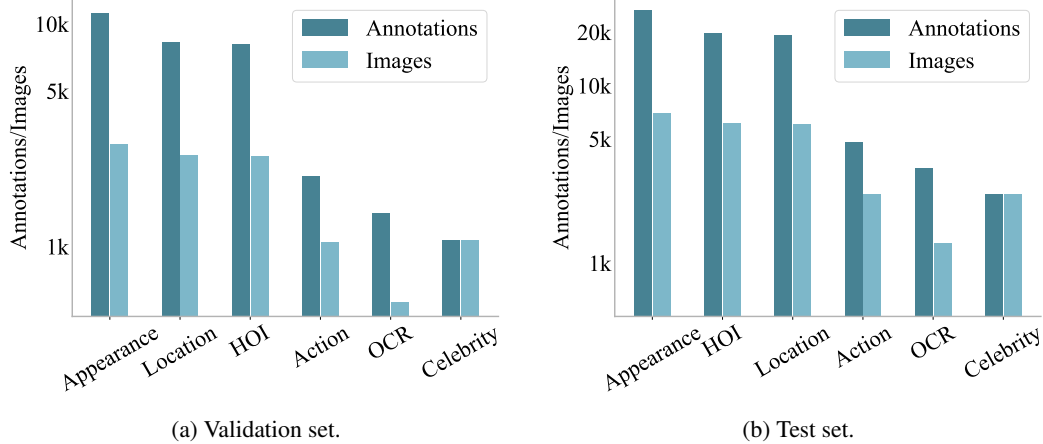

(a) Validation set.

(b) Test set.

Figure 9: The number of annotations and images for each subject in the validation set and the test set.

"man", "child", "jacket", "posture", "group", "event", "image", "stance", "woman", "question", "clothing", "presence", "text", "trousers", "environment", "part" and "sleeves". In Figure 11, we present the 20 most frequently used verbs for each benchmark. In our benchmark, the top 20 verbs are "wearing", "appears", "seems", "sleeved", "holding", "suggesting", "indicating", "suggests", "clad", "paired", "featuring", "located", "stands", "complemented", "indicated", "participating", "depicted", "evidenced", "donned" and "includes".

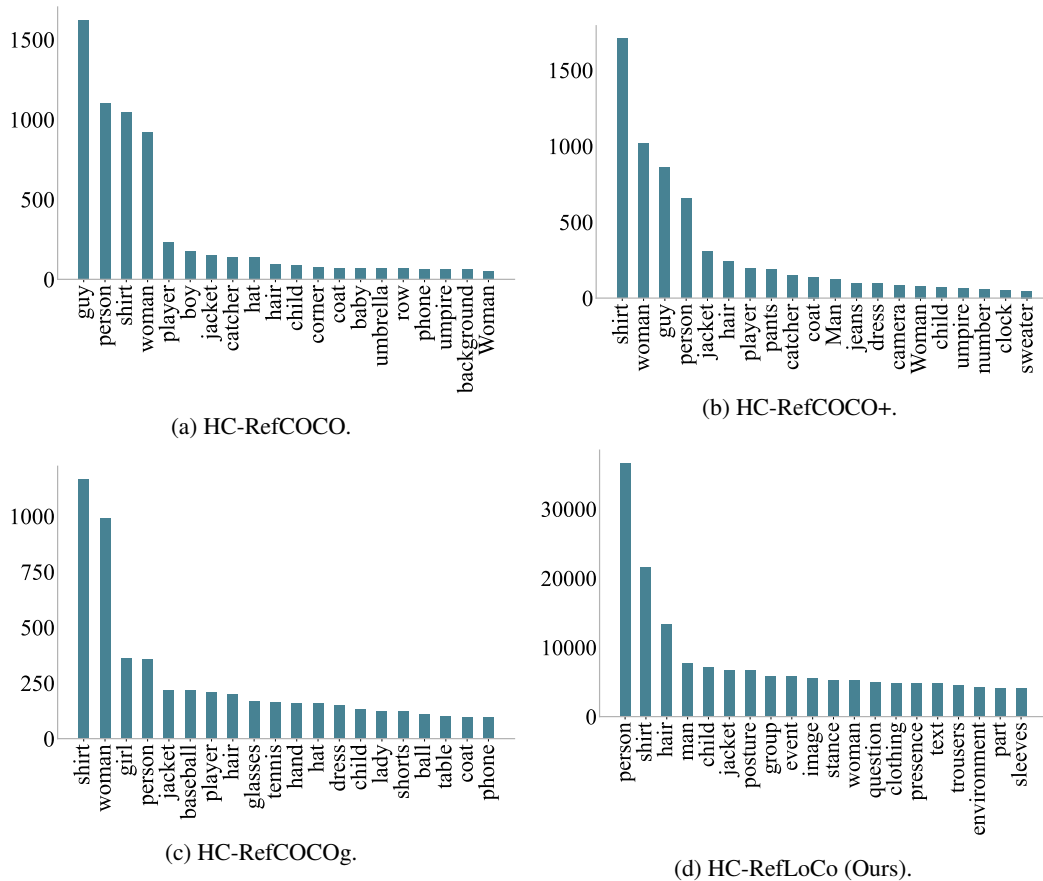

(a) HC-RefCOCO.

(b) HC-RefCOCO+.

(c) HC-RefCOCOg.

(d) HC-RefLoCo (Ours).

Figure 10: The 20 most frequently used nouns in annotations across four different benchmarks.

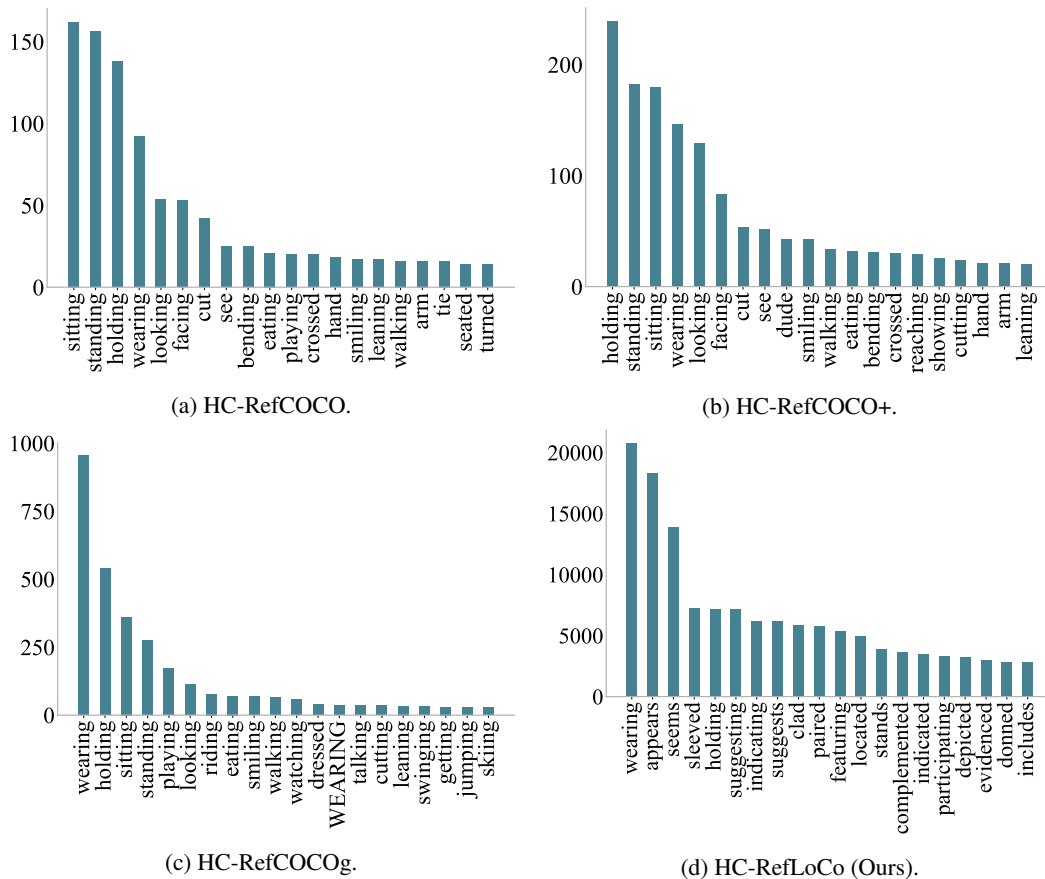

(a) HC-RefCOCO.

(b) HC-RefCOCO+.

(c) HC-RefCOCOg.

(d) HC-RefLoCo (Ours).

Figure 11: The 20 most frequently used verbs in annotations across four different benchmarks.

Table 4: Gender diversity analysis.

| Male | Female | Unrecognizable |
|---|---|---|
| 46.36% | 39.29% | 14.35% |

**Human Diversity.** We use MiVOLO[2] to predict the gender and age of each individual, with subsequent manual corrections. The resulting statistics are presented in Tables 4 and 5, where "unrecognizable" typically refers to cases where faces are either obscured or blurred.

**Scene Diversity.** We begin by obtaining the 365 scene categories from the Places365 benchmark, which is the most extensive dataset for scene recognition. Using GPT-4o, we then consolidate these 365 scenes into 20 broader categories. Each image in our benchmark is subsequently processed by GPT-4o to predict its corresponding scene category, followed by a manual correction of the predictions. The resulting statistics on scene diversity are presented in Table 6. The combined val and test sets are utilized to analyze scene diversity.

# D   Model Cards

Table 7 presents the detailed architecture of each model evaluated in this work.

Table 5: Age diversity analysis.

| Child (0-12) | Adolescence (13–18) | Adult (19–59) | Senior Adult ($\geq$ 60) | Unrecognizable |
|---|---|---|---|---|
| 8.72% | 12.39% | 51.61% | 12.93% | 14.35% |

Table 6: Scene diversity analysis.

| Scene | Percentage |
|---|---|
| Entertainment | 20.53% |
| Sports & Exercise | 15.02% |
| Educational & Cultural Facilities | 8.35% |
| Residential & Domestic Spaces | 8.16% |
| Transportation & Transit | 6.87% |
| Catering & Dining | 6.28% |
| Commercial & Retail Spaces | 5.29% |
| Urban Scenes & Streetscapes | 5.00% |
| Recreational Facilities | 4.00% |
| Outdoor & Adventure | 3.90% |
| Agriculture & Rural | 2.96% |
| Parks & Outdoor Leisure | 2.92% |
| Water & Maritime Scenes | 2.80% |
| Infrastructure & Public Services | 2.59% |
| Industrial & Workplaces | 2.48% |
| Health & Care Facilities | 1.20% |
| Scientific Interest | 0.73% |
| Hospitality, Resorts & Lodging | 0.43% |
| Wildlife | 0.30% |
| Natural Landscapes | 0.18% |

# E   Limitations and Broad Impacts

HC-RefLoCo addresses the limitations of current human-centric REC benchmarks by providing a comprehensive dataset with 13,452 images, 24,129 instances, and 44,738 detailed annotations, averaging 93.2 words each, covering topics such as appearance, human-object interaction, location, action, celebrity, and OCR. However, the benchmark is constrained to only six subjects and the scenes from 13,452 images. Increasing the number of test samples could enhance the credibility and complexity of the evaluation. Additionally, despite meticulous manual reviews, some unintentional annotation errors may still be present. This benchmark is intended solely for positive and constructive purposes in research.

# F   Links and Licenses

**Benchmark Link:** The HC-RefLoCo benchmark is available for download from the Huggingface platform at `https://huggingface.co/datasets/Jinjing713/HC-RefLoCo`.

**Croissant Metadata:** The croissant format metadata for HC-RefLoCo can be accessed at `https://huggingface.co/api/datasets/Jinjing713/HC-RefLoCo/croissant`.

**Code:** The dataloader and evaluation code can be accessed at `https://github.com/ZhaoJingjing713/HC-RefLoCo`.

**DOI:** The DOI of HC-RefLoCo is: 10.57967/hf/2392.

**License:** The HC-RefLoCo dataset is distributed under the Creative Commons Attribution-NonCommercial 4.0 International (CC BY-NC 4.0) license. It is important to note that the images included in the HC-RefLoCo dataset originate from the following sources, each governed by their respective licenses:

Table 7: Architecture of each model. †: a hybrid vision encoder encompassing CLIP-ViT-L/14 [61], CLIP-ConvNeX [61], DINOv2-ViT [57] and Q-Former [92].

| Model | Text Encoder | Vision Encoder |
|---|---|---|
| GPT-4V [54–56] | - | - |
| GroundingGPT [36] | LEGO-7B [36] | CLIP-ViT-L/14 [61] |
| Ferret [88] | Vicuna-7B/13B [8] | CLIP-ViT-L/14 [61] |
| MiniGPT4-v2 [4] | LLaMa 2 Chat-7B [72] | EVA [13] |
| KOSMOS-2 [59] | KOSMOS-1.3B [20] | CLIP-ViT-L/14 [61] |
| Shikra [5] | LLaMA-7B [71] | CLIP-ViT-L/14 [61] |
| OFA [73] | $BART_{Base}$-140M [31] | ResNet50 [18] |
| OFA-Large [73] | $BART_{Large}$-400M [31] | ResNet152 [18] |
| Qwen-VL [3] | Qwen-7B [3] | ViT-bigG [64] |
| Lenna [78] | LLaVA-7B [41] | Swin-L [46] |
| ONE PEACE [74] | Shared Causal Transformer Decoder-4B [74] | |
| SPHINX-MoE [39] | Mixtral-8x7B [21] | Hybrid† |
| SPHINX [39] | LLaMA 2-13B [72] | Hybrid† |
| SPHINX-1k [39] | LLaMA 2-13B [72] | Hybrid† |
| SPHINX-MoE-1k [39] | Mixtral-8x7B [21] | Hybrid† |
| SPHINX-v2-1k [39] | LLaMA 2-13B [72] | Hybrid† |
| PixelLM [98] | LLaVA-7B/13B [41] | CLIP-ViT-L/14 [61] |
| LISA [30] | LLaVA 2-13B [41] | SAM-ViT-H [27] |
| PSALM [96] | Phi 1.5-1.3B [35] | Mask2former-Siwn-B [7] |
| GlaMM [62] | Vicuna-7B [8] | SAM-ViT-H [27] |

